# On-line Learning of Dichotomies

**N. Barkai**
Racah Institute of Physics
The Hebrew University
Jerusalem, Israel 91904
naama@fiz.huji.ac.il

**H. S. Seung**
AT&T Bell Laboratories
Murray Hill, NJ 07974
seung@physics.att.com

**H. Sompolinsky**
Racah Institute of Physics
The Hebrew University
Jerusalem, Israel 91904
and AT&T Bell Laboratories
haim@fiz.huji.ac.il

## Abstract

The performance of on-line algorithms for learning dichotomies is studied. In on-line learning, the number of examples $P$ is equivalent to the learning time, since each example is presented only once. The learning curve, or generalization error as a function of $P$, depends on the schedule at which the learning rate is lowered. For a target that is a perceptron rule, the learning curve of the perceptron algorithm can decrease as fast as $P^{-1}$, if the schedule is optimized. If the target is not realizable by a perceptron, the perceptron algorithm does not generally converge to the solution with lowest generalization error. For the case of unrealizability due to a simple output noise, we propose a new on-line algorithm for a perceptron yielding a learning curve that can approach the optimal generalization error as fast as $P^{-1/2}$. We then generalize the perceptron algorithm to any class of thresholded smooth functions learning a target from that class. For "well-behaved" input distributions, if this algorithm converges to the optimal solution, its learning curve can decrease as fast as $P^{-1}$.

## 1 Introduction

Much work on the theory of learning from examples has focused on *batch* learning, in which the learner is given all examples simultaneously, or is allowed to cycle through them repeatedly. In many situations, it is more natural to consider *on-line* learning paradigms, in which at each time step a new example is chosen. The examples are never recycled, and the learner is not allowed to simply store them (see e.g, Heskes, 1991; Hansen, 1993; Radons, 1993). Stochastic approximation theory (Kushner, 1978) provides a framework for understanding of the local convergence properties of on-line learning of *smooth* functions. This paper addresses the problem of on-line learning of *dichotomies*, for which no similarly complete theory yet exists.

We begin with on-line learning of perceptron rules. Since its introduction in the early 60's, the perceptron algorithm has been used as a simple model of learning a binary classification rule. The algorithm has been proven to converge in finite time and to yield a half plane separating any set of linearly separable examples. The perceptron algorithm, however, is not efficient in the sense of distribution-free PAC learning (Valiant, 1984), for one can construct input distributions that require an arbitrarily long convergence time. In a recent paper (Baum, 1990) Baum proved that the perceptron algorithm applied in an on-line mode, converges as $P^{-1/3}$ when learning a half space under a uniform input distribution, where $P$ is the number of presented examples drawn at random. For on-line learning $P$ is also the number of time steps. Baum also generalized his result to any "non-malicious" distribution. Kabashima has found the same power law for learning a two-layer parity machine with non-overlapping inputs, using an on-line least action algorithm (Kabashima, 1994).

If efficiency is measured only by the number of examples used (disregarding time), these particular on-line algorithms are much worse than batch algorithms. Any batch algorithm which is able to correctly classify a given set of $P$ examples will converge as $P^{-1}$ (Vapnik, 1982; Amari, 1992; Seung, 1992). In this paper, we construct on-line algorithms that can actually achieve the same power law as batch algorithms, demonstrating that the results of Baum and Kabashima do not reflect a fundamental limitation of on-line learning.

In Section 3, we study on-line algorithms for perceptron learning of a target rule that is not realizable by a perceptron. Here it is nontrivial to construct an algorithm that even converges to the optimal one, let alone to optimize the rate of convergence. For the special case of a target rule that is a perceptron corrupted by output noise this can be done. In Section 4, our results are generalized to dichotomies generated by thresholding smooth functions. In Section 5 we summarize the results.

## 2    On-line learning of a perceptron rule

We consider a half space rule generated by a normalized teacher perceptron $\mathbf{W}_0 \in R^N$, $\mathbf{W}_0 \cdot \mathbf{W}_0 = 1$ such that any vector $\mathbf{S} \in R^N$ is given a label $\sigma_0(\mathbf{S}) = \text{sgn}(\mathbf{W}_0 \cdot \mathbf{S})$. We study the case of a Gaussian input distribution centered at zero with a unit variance in each direction in space:

$$P(\mathbf{S}) = \prod_{i=1}^{N} \frac{1}{\sqrt{2\pi}} e^{-S_i^2/2} \tag{1}$$

Averages over this input distribution will be written with angle brackets $\langle\rangle$. A student perceptron $\mathbf{W}$ is trained by an on-line perceptron algorithm. At each time step, an input $\mathbf{S} \in R^N$ is drawn at random, according to distribution Eq. (1) and the student's output $\sigma(\mathbf{S}) = \text{sgn}(\mathbf{W} \cdot \mathbf{S})$ is calculated. The student is then updated according to the perceptron rule:

$$\mathbf{W}' = \mathbf{W} + \frac{\eta}{N}\epsilon(\mathbf{S}; \mathbf{W})\sigma_0(\mathbf{S})\mathbf{S} \tag{2}$$

and is then normalized so that $\mathbf{W} \cdot \mathbf{W} = 1$ at all times. The factor $\epsilon(\mathbf{S}; \mathbf{W})$ denotes the error of the student perceptron on the input $\mathbf{S}$: $\epsilon = 1$ if $\sigma(\mathbf{S})\sigma_0(\mathbf{S}) = 1$, and 0 otherwise. The *learning rate* $\eta$ is the magnitude of change of the weights at each time step. It is scaled by $N$ to ensure that the change in the overlap $R = \mathbf{W} \cdot \mathbf{W}_0$ is of order $1/N$. Thus, a change of $\mathcal{O}(1)$ occurs only after presentation of $P = \mathcal{O}(N)$ examples.

The performance of the student is measured by the generalization error, defined as the probability of disagreement between the student and the teacher on an arbitrary input $\epsilon_g = \langle\epsilon(\mathbf{S}; \mathbf{W})\rangle$. In the present case, $\epsilon_g$ is

$$\epsilon_g = \frac{\cos^{-1} R}{\pi}. \tag{3}$$

Although for simplicity we analyze below the performance of the perceptron rule (2) only for large $N$, our results apply to finite $N$ as well. Multiplying Eq. (2) by $\mathbf{W}_0$ after incorporation of the normalization operation and averaging with respect to the input distribution (1), yields the following differential equation for $R(\alpha)$ where $\alpha = P/N$,

$$\frac{dR}{d\alpha} = \eta\frac{1 - R^2}{\sqrt{2\pi}} - \eta^2\frac{R\cos^{-1} R}{2\pi}. \tag{4}$$

Here terms of order $\sqrt{\eta/N}$ have been neglected.

The evolution of the overlap $R$, and thus of the generalization error, depends on the schedule at which the learning rate $\eta$ decreases. We consider two cases, a constant $\eta$ and a time-dependent $\eta$.

**Constant learning rate**: When $\eta$ is held fixed, Eq. (4) has a stable fixed point at $R < 1$, and hence $\epsilon_g$ converges to an $\eta$-dependent nonzero value $\epsilon_\infty(\eta)$. For $\eta \ll 1$, $1 - R_\infty(\eta) \propto \eta^2$ and $\epsilon_g \propto \sqrt{1 - R}$ is therefore proportional to $\eta$,

$$\epsilon_\infty(\eta) = \eta/\sqrt{2\pi^3} \ . \tag{5}$$

The convergence to this value is exponential in $\alpha$, $\epsilon_g(\alpha) - \epsilon_\infty(\eta) \sim \exp(-\eta\alpha/\sqrt{2\pi})$.

**Time-dependent learning rate**: Convergence to $\epsilon_g = 0$ can be achieved if $\eta$ decreases slowly enough with $\alpha$. We study the limiting behaviour of the system for $\eta$ which is decreasing with time as $\eta = \left(\eta_0\sqrt{2\pi}\right)\alpha^{-z}$.

$z > 1$. In this case the rate is reduced too fast before a sufficient number of examples have been seen. This results in $R$ which does not converge to 1 but instead to a smaller value that depends on its initial value.

$z < 1$. The system follows the change in $\eta$ adiabatically. Hence, to first order in $\alpha^{-1}$, $\epsilon_g(\alpha) = \epsilon_\infty(\eta(\alpha))$. Thus, $\epsilon_g$ converges to zero with an asymptotic rate $\epsilon_g(\alpha) \sim \alpha^{-z}$.

$z = 1$. The behaviour of the system depends on the prefactor $\eta_0$:

$$
\begin{aligned}
\epsilon_g \ &\sim \ \left(\frac{1}{\pi}\frac{\eta_0^2}{\eta_0 - 1}\right)\frac{1}{\alpha} \qquad \eta_0 > 1 \\
&\sim \ \frac{\sqrt{\log\alpha}}{\alpha} \qquad\qquad \eta_0 = 1 \\
&\sim \ \frac{A}{\alpha^{\eta_0}} \qquad\qquad\quad \eta_0 < 1
\end{aligned}
\tag{6}
$$

where $A$ depends on the initial condition. Thus the optimal asymptotic change of $\eta$ is $2\sqrt{2\pi}/\alpha$, in which case the error will behave asymptotically as $\epsilon_g(\alpha) \sim 1.27/\alpha$. This is not far from the batch asymptotic (Seung, 1992) $\epsilon_g(\alpha) \sim 0.625/\alpha$. We have confirmed these results by numerical simulation of the algorithm Eq. (2). Figure 1 presents the results of the optimal learning schedule, i.e., $\eta = 2\sqrt{2\pi}/\alpha$. The numerical results are in excellent agreement with the prediction $\epsilon_g(\alpha) = 1.27/\alpha$ for the asymptotic behavior. Finally, we note that our analysis of the time-dependent case is similar to that of Kabashima and Shinomoto for a different on-line learning problem (Kabashima, 1993).

## 3 On-line learning of a perceptron with output noise

In the case discussed above, the task can be fully realized by a perceptron, i.e., there is a perceptron $\mathbf{W}$ such that $\epsilon_g = 0$. In more realistic situations a perceptron will only provide an approximation of the target function, so that the minimal value of $\epsilon_g$ is greater than zero. These cases are called *unrealizable tasks*. A drawback of the above on-line algorithm is that, for a general unrealizable task, it does not converge to the optimal perceptron, i.e., it does not approach the minimum of $\epsilon_g$. To illustrate this fact we consider a perceptron rule corrupted by output noise. The label of an input $\mathbf{S}$ is $\sigma_0(\mathbf{S})$, where $\sigma_0(\mathbf{S}) = \text{sgn}(\mathbf{W}_0 \cdot \mathbf{S})$ with probability $1 - p$, and $-\text{sgn}(\mathbf{W}_0 \cdot \mathbf{S})$ with probability $p$. We assume $0 \leq p \leq 1/2$. For reasons which will become clear later, the input distribution is taken as a Gaussian centered at $\mathbf{U}$

$$P(\mathbf{S}) = \prod_{i=1}^{N} \frac{1}{\sqrt{2\pi}} e^{-(S_i - U_i)^2/2} \tag{7}$$

In this case $\epsilon_g$ is given by

$$\epsilon_g = p + \left(\int_{-\infty}^{-q} Dy\, H\left(\frac{-q_0 - Ry}{\sqrt{1 - R^2}}\right) + \int_{-q}^{\infty} Dy\, H\left(\frac{q_0 + Ry}{\sqrt{1 - R^2}}\right)\right) . \tag{8}$$

where $q_0 = \mathbf{U} \cdot \mathbf{W}_0$ denotes the overlap between the center of the distribution and the teacher perceptron, and $q = \mathbf{U} \cdot \mathbf{W}$ is the overlap between the center of the distribution and $\mathbf{W}$. The integrals in Eq. (8) are

with respect to a Gaussian measure $Dy = \exp(-y^2/2)/\sqrt{2\pi}$ and $H(x) = \int_x^\infty Dy$. Note that the optimal perceptron is the teacher $\mathbf{W} = \mathbf{W}_0$ i.e., $R = 1, q = q_0$, which yields the minimal error $\epsilon_{min} = p$.

First, we consider training with the normalized perceptron rule (2). In this case, we obtain differential equations for two variables: $R$ and $q$. Solving these equations we find that in general, $\mathbf{W}$ converges to a vector with a direction which is in the plane of $\mathbf{W}_0$ and $\mathbf{U}$ and is does not point in the direction of $\mathbf{W}_0$ even in the limit of $\eta \to 0$. Here we present the result for the limit of $\eta \to 0$ and small noise level, i.e., $p \ll 1$. In this case, we obtain for $\epsilon_\infty(\eta = 0)$

$$\epsilon_\infty(0) = p + p\frac{(1 - 2H(q_0))\sqrt{u^2 - q_0^2}}{1 + (u^2 - q_0^2)} + \mathcal{O}(p^2) \tag{9}$$

where $u = |\mathbf{U}|$ is the magnitude of the center of the input distribution. For $p = 0$, the only solution is $R = 1$ and $q = q_0$, in agreement with the previous results. For $p > 0$ the optimal solution is retrieved only in the following special cases: (i) the input distribution is isotropic, i.e., $q_0 = u = 0$; (ii) when $\mathbf{U}$ is parallel to $\mathbf{W}_0$, i.e., $u = q_0$; and (iii) when $\mathbf{U}$ is orthogonal to $\mathbf{W}_0$, i.e., $q_0 = 0$. This holds also for large value of $p$. In these special cases, the symmetry of the input distribution relative to the teacher vector, guarantees that the deviations from $\mathbf{W} = \mathbf{W}_0$ incurred by the inputs that come with the wrong label cancel each other on average. According to Eq. (9), for other directions of $\mathbf{U}$, $\epsilon_g$ is above the optimal value. Note that the additional term in $\epsilon_g$ is of the same order of magnitude ($\mathcal{O}(p)$) as the minimal error.

In the following we suggest a modified on-line algorithm for learning a perceptron rule with output noise. The student weights are changed according to

$$\mathbf{W}' = \mathbf{W} + \frac{\eta}{N}\epsilon(\mathbf{S}; \mathbf{W})\sigma_0(\mathbf{S})(\mathbf{S} - \mathbf{T}(\mathbf{S})) \tag{10}$$

followed by a normalization of $\mathbf{W}$. This algorithm differs from the perceptron algorithm in that the change in $\mathbf{W}$ is not proportional to the present input, but to a shifted vector. The shifting vector $\mathbf{T}(\mathbf{S})$, is determined by the requirement that the teacher $\mathbf{W}_0$ will be a fixed point of the algorithm in the limit of $\eta \to 0$. This is equivalent to the condition

$$\langle \epsilon_0(\mathbf{S})\sigma_0(\mathbf{S})(\mathbf{S} - \mathbf{T}(\mathbf{S})) \rangle = 0 \tag{11}$$

where $\epsilon_0(\mathbf{S})$ is the error function for $\mathbf{S}$ when $\mathbf{W} = \mathbf{W}_0$. This condition does not determine $\mathbf{T}$ uniquely. A simple choice is one for which $\mathbf{T}$ is independent of $\mathbf{S}$. This leads to

$$\mathbf{T} = \frac{\langle \mathrm{sgn}(\mathbf{W}_0 \cdot \mathbf{S})\mathbf{S} \rangle}{\langle \mathrm{sgn}(\mathbf{W}_0 \cdot \mathbf{S}) \rangle} = \frac{\langle \sigma_0 \mathbf{S} \rangle}{\langle \sigma_0 \rangle} \tag{12}$$

where we used the fact that for any $\mathbf{S}$, $\epsilon_0(\mathbf{S})\sigma_0(\mathbf{S})$ equals $-\mathrm{sgn}(\mathbf{W}_0 \cdot \mathbf{S})$ with probability $p$, and zero with probability $(1 - p)$. This uniform shift is possible only when $\langle \sigma_0 \rangle \neq 0$, namely when the average frequencies of $+1$ and $-1$ labels are not equal. If this is not the case, one has to choose nonuniform forms of $\mathbf{T}(\mathbf{S})$. Note that in general $\mathbf{T}$ has to be learned so that Eq. (10) has to be supplemented by appropriate equations for changing $\mathbf{T}$. In the case of Eq. (12), one can easily learn separately the numerator and denominator by running averages of $\sigma_0\mathbf{S}$ and $\sigma_0$, respectively. We have studied analytically the above algorithm for the case of the Gaussian input distribution Eq. (7), in the limit of large $N$. The shifting vector is given by

$$\mathbf{T} = \mathbf{U} + \mathbf{W}_0\sqrt{\frac{2}{\pi}}\frac{\exp(-q_0^2/2)}{1 - 2H(q_0)} \tag{13}$$

The differential equations for the overlaps $R$ and $q$ in the neighborhood of the point $R = 1$ and $q = q_0$ are,

$$\frac{d\delta R}{d\alpha} = -\eta\sqrt{2/\pi}\exp(-q_0^2/2)\delta R + \frac{1}{2}\eta^2 p \tag{14}$$

$$\frac{d\delta q}{d\alpha} = -\eta\frac{\exp(-q_0^2/2)}{\sqrt{2\pi}}(\delta q + q_0\delta R) + \frac{1}{2}\eta^2 q_0 p$$

where $\delta R = 1 - R$ and $\delta q = q_0 - q$. In the limit $\eta \to 0$, $R = 1$ and $q = q_0$ is indeed a stable fixed point of the algorithm, so that the student converges to the optimal perceptron $\mathbf{W}_0$, and hence the generalization error converges to its minimal value $\epsilon_{min} = p$. Since, unlike Eq. (4), the coefficient of the $\eta^2$ term in Eq.

(14) is constant, $\delta R_\infty(\eta) \propto \eta$, for small fixed $\eta$, and not to $\eta^2$. Thus, in this case, the generalization error approaches, in the limit $\alpha \to \infty$, the value

$$\epsilon_\infty(\eta) = p + \sqrt{\eta p} \frac{\exp(-q_0^2/4)}{(2\pi^3)^{1/4}} \ . \tag{15}$$

For a time-dependent $\eta$, the convergence to the optimal weights depends on the choice of $\eta(n)$, as in the case of the noiseless perceptron rule. For $\eta = \left(\eta_0 \sqrt{\pi/2} \exp(q_0^2/2)\right) \alpha^{-z}$, with $z \le 1$, the error converges to $p$. For $z < 1$, to first order in $1/\alpha$, $\epsilon_g(\alpha) = \epsilon_\infty(\eta(\alpha))$, yielding

$$\epsilon_g(\alpha) - p \sim \alpha^{-z/2}. \tag{16}$$

When $z = 1$, the rate of convergence depends on the value of $\eta_0$.

$$\epsilon_g(\alpha) - p \sim \begin{cases} \alpha^{-1/2} & \eta_0 > 1 \\ \alpha^{-\eta_0/2}, & \eta_0 < 1 \end{cases} \tag{17}$$

and logarithmic corrections to $\alpha^{-1/2}$ for $\eta_0 = 1$. Thus, the optimal rate of convergence is

$$\epsilon_g(\alpha) - p \approx \sqrt{\frac{2p}{\pi\alpha}} \tag{18}$$

which is achieved for $\eta_0 = 2$.

We have tested successfully this algorithm by simulations of learning a perceptron rule with output noise with several input distributions, including the Gaussian, of Eq. (7). Figure 2 presents the generalization error as a function of $\alpha$ for the Gaussian distribution, with $p = 0.2$, and we have chosen $\eta_0 = 2$. The error converges to the optimal value 0.2 as $\alpha^{-1/2}$ in agreement with the theory. For comparison the result of the usual perceptron algorithm is also presented. This algorithm converges to $\epsilon_g \approx 0.32$, clearly larger than the optimal value.

## 4  On-line learning of thresholded smooth functions

Our results for the realizable perceptron can be extended to a more general class of dichotomies, namely thresholded smooth functions. They are defined as dichotomies of the form

$$\sigma(\mathbf{S}; \mathbf{W}) = \text{sgn}(f(\mathbf{S}; \mathbf{W})) \tag{19}$$

where $f$ is a differentiable function of a set of parameters, denoted by $\mathbf{W}$, and $\mathbf{S}$ is the input vector. We consider here the case of a realizable task, where the examples are given with labels $\sigma_0$ corresponding to a target machine $\mathbf{W}_0$ which is in the $\mathbf{W}$ space. For this task we propose the following generalization of the perceptron rule (2)

$$\mathbf{W}' = \mathbf{W} + \eta\epsilon(\mathbf{S}; \mathbf{W})\sigma_0(\mathbf{S})\nabla f(\mathbf{S}; \mathbf{W}) \tag{20}$$

where $\nabla$ denotes a gradient w.r.t. $\mathbf{W}$. Then, as we argue below, the vector $\mathbf{W}_0$ is a stable fixed point in the limit of $\eta \to 0$. Furthermore, for constant small $\eta$ the residual error scales as $\epsilon_\infty \propto \eta$. For $\eta \sim \alpha^{-z}$, $z < 1$, $\epsilon_g(\alpha) \sim \epsilon_\infty(\eta(\alpha)) \sim \alpha^{-z}$.

To show this, let us consider for simplicity the one-dimensional case, $w' = w + \eta g(w, s)$, where

$$g(w, s) = \theta(-f(w, s)f(w_0, s)) \, \text{sgn}(f(w_0, s)) \frac{\partial f}{\partial w} \ . \tag{21}$$

This equation can be converted into a Markov equation for the probability distribution, $P(w, n)$ (Van Kampen, 1981)

$$P(w, n+1) = \int dw' W(w'|w) P(w', n) \tag{22}$$

where $W(w|w') = < \delta(w' - w - \eta g(w, s)) >$ is the transition rate from w to w'. In the limit of small fixed $\eta$, the equilibrium distribution, $P_\infty$, can be shown to have the following scaling form,

$$P_\infty(w; \eta) = \frac{1}{\eta} F(\delta w/\eta) \tag{23}$$

where $\delta w = w - w_0$ and $F(x)$ obeys the following difference equation

$$\hat{L}F(x) \equiv \sum_{\sigma=\pm1} \theta((f_0' + \sigma x)f_0')|(f_0' + \sigma x)|F(x + \sigma f_0') - |x|F(x) = 0 \qquad (24)$$

where $f_0'$ is the value of the gradient $\partial f(w_0, s)/\partial w$ at the decision boundary of $f(w_0, s)$, namely at the point $s$ obeying $f(w_0, s) = 0$. Note that since we are interested in normalizable solutions of Eq. (24), $F(x)$ has to vanish for for all $x > |f_0'|$. This result is valid provided the input distribution is smooth and nonvanishing near the decision boundary. Furthermore, $\partial f/\partial w$ at $w_0$ may not vanish on the decision boundary. Under the same conditions, it can be shown that the error is homogeneous in $\delta w$ with degree 1, hence it should scale linearly with $\eta$, i.e., $\epsilon_\infty \propto \eta$. It should be noted that, unlike other on-line learning problems (Heskes, 1991; Hansen, 1993; Radons, 1993), the equilibrium distribution is our case is not Gaussian.

For a time-dependent $\eta$ of the form $\eta = \eta_0 n^{-z}$, $z < 1$, $P(w, n)$ at long times is of the form

$$P(w, n) = \frac{1}{\eta(n)} \left( F(\delta w/\eta(n)) + \frac{G(\delta w/\eta(n))}{n^{1-z}} \right) \qquad (25)$$

where $F$ is the stationary distribution, given by Eq. (24) and the coefficient of the correction, $G$, solves the inhomogeneous equation

$$zx\frac{dF}{dx} + zF(x) = \eta_0\hat{L}G(x) \qquad (26)$$

where the linear operator $\hat{L}$ is defined in Eq. (24). Thus, to leading order in inverse time, the system follows adiabatically the finite-$\eta$ stationary distribution, yielding $\epsilon_g(n)$ which vanishes asymptotically as $\epsilon_g(n) \propto \eta(n) \sim n^{-z}$. The optimal schedule is obtained for $z = 1$. In this case, $P(w, n) = \eta^{-1}(n)F(\delta w/\eta(n))$ where $F(x)$ solves the homogeneous equation

$$zx\frac{dF}{dx} + zF(x) = \eta_0\hat{L}F(x) \qquad (27)$$

For sufficiently large $\eta_0$, this equation has a solution, implying that $\epsilon_g \propto n^{-1}$.

Similarly, the results of Section 3 can also be extended to the case of thresholded- smooth functions with a probability $p$ of an error due to isotropic output noise. In this case, the optimal choice is again $\eta \propto n^{-1}$ yielding $\epsilon_g - p \approx \sqrt{\eta}$. It should be noted that for this case, the probability distribution for small $\eta$ does reduce to a Gaussian distribution in $\delta w/\sqrt{\eta}$. Using a multidimensional Markov equation, it is straightforward to extend these results to higher dimensions. The small $\eta$ limit yields equations similar to Eqs. (24-26), that involve integration over the decision boundary of $f(\mathbf{W}, \mathbf{S})$.

## 5   Summary and Discussion

We have found that the perceptron rule (2) with normalization can lead to a variety of learning curves, depending on the schedule at which the learning rate is decreased. The optimal schedule leads to an inverse power law learning curve, $\epsilon_g \sim \alpha^{-1}$. Baum's results (Baum, 1990) of a non-normalized perceptron with a constant learning rate can be viewed as a special case of the above analysis. In the non-normalized perceptron algorithm, the magnitude of the student's weights grow with $\alpha$ as $|\mathbf{W}| \sim \alpha^{1/3}$. The time evolution of the overlap $R$, and thus of the generalization error is governed by the effective learning rate $\eta_{\text{eff}} = \eta/|\mathbf{W}|$ leading via Eq. (6) to the result $\epsilon_g \sim \alpha^{-1/3}$. Similar results apply to the two-layer parity machine studied in (Kabashima, 1994).

Our analysis, leading to the equations of motion (4) and (14), was based on the limit of large $N$ and $P$, such that $\alpha = P/N$ remains finite. We would like to stress however, that this limit is only necessary in deriving the full form of the learning curve, i.e., $R(\alpha)$ for all $\alpha$. On the other hand, our results for the large $P$ asymptote of the learning curve for small $\eta$ are valid for finite $N$ as well, as implied by the general treatment of the previous section.

Unrealizable perceptron rules present a more complicated problem. We have presented here a modified perceptron algorithm that converges to the optimal solution in the special case of an isotropic output noise.

In this case, the convergence to the optimal error is as $\alpha^{-1/2}$. This is the same power law as obtained in the standard sample complexity upper bounds (Vapnik, 1982) and in the approximate replica symmetric calculations (Seung, 1992) for batch learning of unrealizable rules. It should be stressed however, that the success of the modified algorithm in the case of an output noise depends on the fact that the errors made by the optimal solution are uncorrelated with the input. Thus, finding an on-line algorithm that can cope with other types of unrealizability remains an important problem.

The learning algorithms for the perceptron rule, without and with output noise, can be generalized to learning thresholded smooth functions, assuming certain reasonable properties of the input distribution are present, as shown in Section 4. The dependence of the learning curve on the learning rate schedule remains roughly the same as in the perceptron case. This implies that on-line learning of realizable dichotomies, with possible output noise, can achieve the same power laws in the number of examples that is typical of batch learning of such rules. Furthermore, the on-line formulation possesses the theoretical virtues of addressing time as well as sample complexity, so that the same power laws imply the polynomial relationship between the time and the achieved error level. The above conclusions assume that the equilibrium state at small learning rates is unique, which in general is not the case. The issue of overcoming local minima in on-line learning is a difficult problem (Heskes, 1992) Finally, the theoretical results for on-line learning has the important advantage of not requiring the use of the often problematic replica formalism.

## Acknowledgements

We are grateful for helpful discussions with Y. Freund, M. Kearns, R. Schapire, and E. Shamir, and thank Y. Kabashima for bringing his paper to our attention. HS is partially supported by the Fund for Basic Research of the Israeli Academy of Arts and Sciences.

## References

S. Amari, N. Fujita, and S. Shinomoto. Four types of learning curves. *Neural Comput.*, 4:605–618, 1992.

E. B. Baum. The perceptron algorithm is fast for nonmalicious distributions. *Neural Comput.*, 2:248–260, 1990.

H. J. Kushner and D. S. Clark. *Stochastic approximation methods for constrained and unconstrained systems.* Springer, Berlin, 1978.

L. K. Hansen, R. Pathria, and P. Salamon. Stochastic dynamics of supervised learning. *J. Phys.*, A26:63–71, 1993.

T. Heskes and B. Kappen. Learning processes in neural networks. *Phys. Rev.*, A44:2718–2762, 1991.

T. Heskes, E. T. P. Slijpen, and B. Kappen. Learning in neural networks with local minima. *Phys. Rev.*, A46:5221–5231, 1992.

Y. Kabashima. Perfect loss of generalization due to noise in $k = 2$ parity machines. *J. Phys.*, A27:1917–1927, 1994.

Y. Kabashima and S. Shinomoto. Incremental learning with and without queries in binary choice problems. In *Proc. of IJCNN*, 1993.

G. Radons. On stochastic dynamics of supervised learning. *J. Phys.*, A26:3455–3461, 1993.

H. S. Seung, H. Sompolinsky, and N. Tishby. Statistical mechanics of learning from examples. *Phys. Rev.*, A45:6056–6091, 1992.

L. G. Valiant. A theory of the learnable. *Commun. ACM*, 27:1134–1142, 1984.

N. G. Van Kampen. *Stochastic processes in physics and chemistry.* North holland 1981.

V. N. Vapnik. *Estimation of Dependences based on Empirical Data.* Springer-Verlag, New York, 1982.

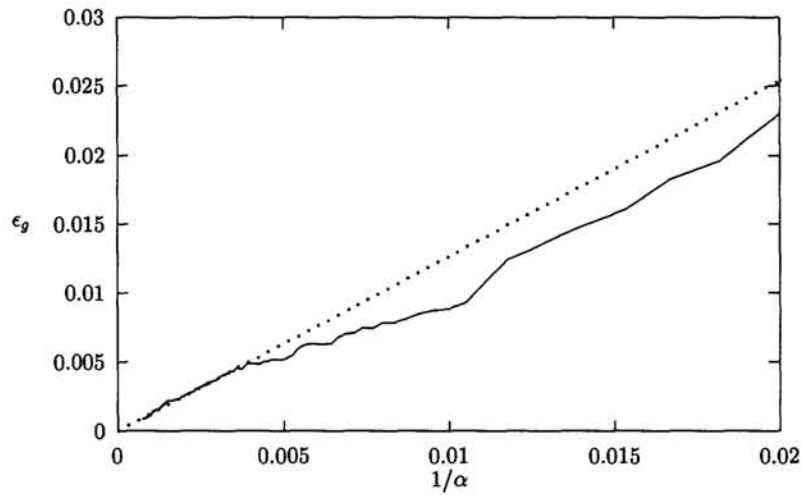

Figure 1: Asymptotic performance of a realizable perceptron. Simulation results for $\eta_0 = 2$ and $N = 50$ (solid curve) are compared with the theoretical prediction $\epsilon_g = 1.27/\alpha$ (dashed curve).

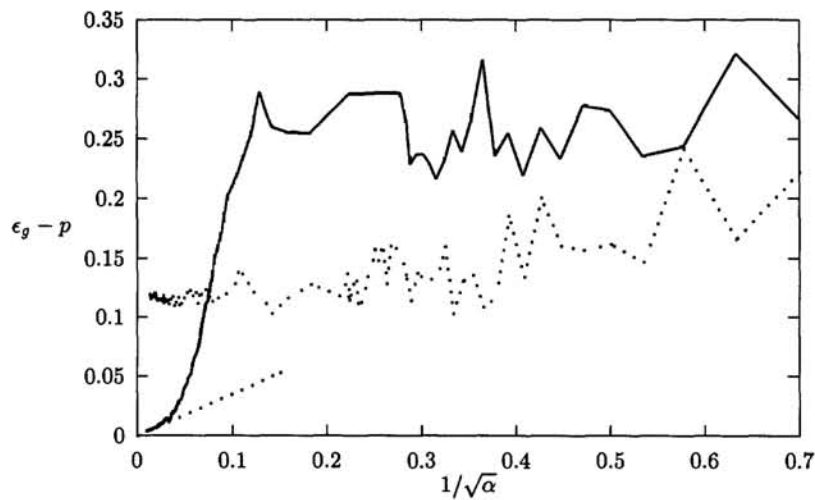

Figure 2: Simulation results for on-line learning of a perceptron with output noise. Here $\eta_0 = 2$, $p = 0.2$, $N = 250$, $u = 4$, and $q_0 = -1.95$. The regular perceptron learning (dashed curve) is compared with the modified algorithm (solid curve). The dashed line shows the theoretical prediction Eq. (18)